# Connected Letter Recognition with a Multi-State Time Delay Neural Network

**Hermann Hild**   and   **Alex Waibel**
School of Computer Science
Carnegie Mellon University
Pittsburgh, PA 15213-3891, USA

## Abstract

The Multi-State Time Delay Neural Network (MS-TDNN) integrates a nonlinear time alignment procedure (DTW) and the high-accuracy phoneme spotting capabilities of a TDNN into a connectionist speech recognition system with word-level classification and error backpropagation. We present an MS-TDNN for recognizing continuously spelled letters, a task characterized by a small but highly confusable vocabulary. Our MS-TDNN achieves 98.5/92.0% word accuracy on speaker dependent/independent tasks, outperforming previously reported results on the same databases. We propose training techniques aimed at improving sentence level performance, including free alignment across word boundaries, word duration modeling and error backpropagation on the sentence rather than the word level. Architectures integrating submodules specialized on a subset of speakers achieved further improvements.

## 1   INTRODUCTION

The recognition of spelled strings of letters is essential for all applications involving proper names, addresses or other large sets of special words which due to their sheer size can not be in the basic vocabulary of a recognizer. The high confusability of the English letters makes the seemingly easy task a very challenging one, currently only addressed by a few systems, e.g. those of R. Cole et. al. [JFC90, FC90, CFGJ91] for isolated spoken letter recognition. Their connectionist systems first find a broad phonetic segmentation, from which a letter segmentation is derived, which is then

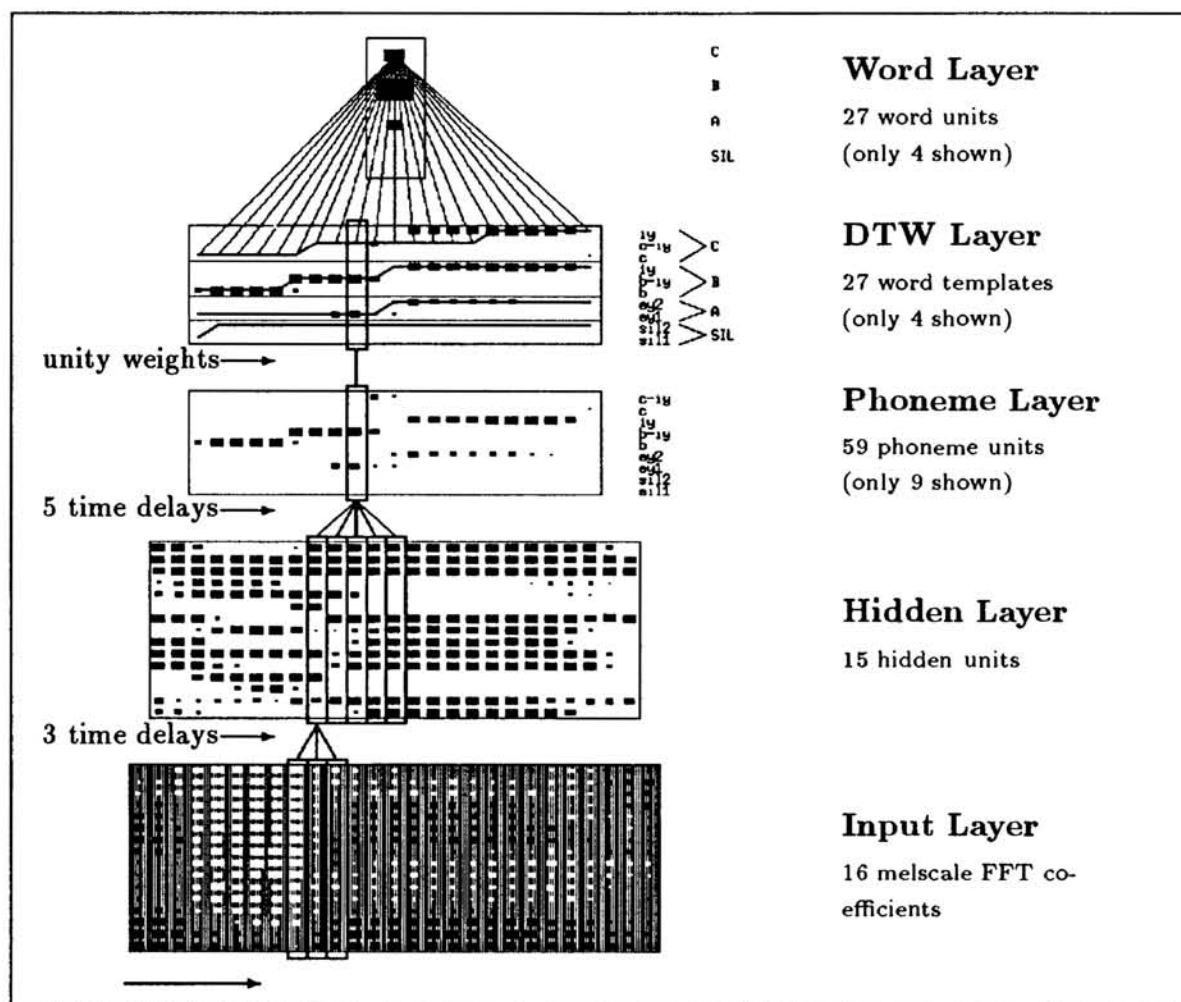

Figure 1: The MS-TDNN recognizing the excerpted word 'B'. Only the activations for the words 'SIL', 'A', 'B', and 'C' are shown.

classified by another network.    In this paper, we present the MS-TDNN as a connectionist speech recognition system for connected letter recognition. After describing the baseline architecture, training techniques aimed at improving sentence level performance and architectures with gender-specific subnets are introduced.

**Baseline Architecture.** Time Delay Neural Networks (TDNNs) can combine the robustness and discriminative power of Neural Nets with a time-shift invariant architecture to form high accuracy phoneme classifiers [WHH+89]. The Multi-State TDNN (MS-TDNN) [HFW91, Haf92, HW92], an extension of the TDNN, is capable of classifying words (represented as sequences of phonemes) by integrating a nonlinear time alignment procedure (DTW) into the TDNN architecture. Figure 1 shows an MS-TDNN in the process of recognizing the excerpted word 'B', represented by 16 melscale FFT coefficients at a 10-msec frame rate. The first three layers constitute a standard TDNN, which uses sliding windows with time delayed connections to compute a score for each phoneme (state) for every frame, these are the activations in the "Phoneme Layer". In the "DTW Layer", each word to be recognized is modeled by a sequence of phonemes. The corresponding activations are simply

copied from the Phoneme Layer into the word models of the DTW Layer, where an optimal alignment path is found for each word. The activations along these paths are then collected in the word output units. All units in the DTW and Word Layer are linear and have no biases. 15 (25 to 100) hidden units per frame were used for speaker-dependent (-independent) experiments, the entire 26 letter network has approximately 5200 (8600 to 34500) parameters.

**Training** starts with "bootstrapping", during which only the front-end TDNN is used with fixed phoneme boundaries as targets. In a second phase, training is performed with word level targets. Phoneme boundaries are freely aligned within given word boundaries in the DTW layer. The error derivatives are backpropagated from the word units through the alignment path and the front-end TDNN.

The choice of sensible objective functions is of great importance. Let $Y = (y_1, \ldots, y_n)$ the output and $T = (t_1, \ldots, t_n)$ the target vector. For training on the phoneme level (bootstrapping), there is a target vector $T$ for each frame in time, representing the correct phoneme $j$ in a "1-out-of-$n$" coding, i.e. $t_i = \delta_{ij}$. To see why the standard *Mean Square Error* $(MSE = \sum_{i=1}^{n} (y_i - t_i)^2)$ is problematic for "1-out-of-$n$" codings for large $n$ ($n = 59$ in our case), consider for example that for a target $(1.0, 0.0, \ldots, 0.0)$ the output vector $(0.0, \ldots, 0.0)$ has only half the error than the more desirable output $(1.0, 0.2, \ldots, 0.2)$. To avoid this problem, we are using

$$E_{McClelland}(T, Y) = \sum_{i=1}^{n} log(1 - (y_i - t_i)^2)$$

which (like cross entropy) punishes "outliers" with an error approaching infinity for $|t_i - y_i|$ approaching 1.0. For the word level training, we have achieved best results with an objective function similar to the "Classification Figure of Merit (CFM)" [HW90], which tries to maximize the distance $d = y_c - y_{hi}$ between the correct score $y_c$ and the highest incorrect score $y_{hi}$ instead of using absolute target values of 1.0 and 0.0 for correct and incorrect word units:

$$E_{CFM}(T, Y) = f(y_c - y_{hi}) = f(d) = (1 - d)^2$$

The philosophy here is not to "touch" any output unit not directly related to correct classification. We found it even useful to apply error backpropagation only in the case of a wrong or too narrow classification, i.e. if $y_c - y_{hi} < \delta_{safety\_margin}$.

## 2    IMPROVING CONTINUOUS RECOGNITION

### 2.1    TRAINING ACROSS WORD BOUNDARIES

A proper treatment of word[1] boundaries is especially important for a short word vocabulary, since most phones are at word boundaries. While the phoneme boundaries within a word are freely aligned by the DTW during "word level training", the word boundaries are fixed and might be error prone or suboptimal. By extending the alignment one phoneme to the left (last phoneme of previous word) and the right (first phoneme of next word), the word boundaries can be optimally adjusted

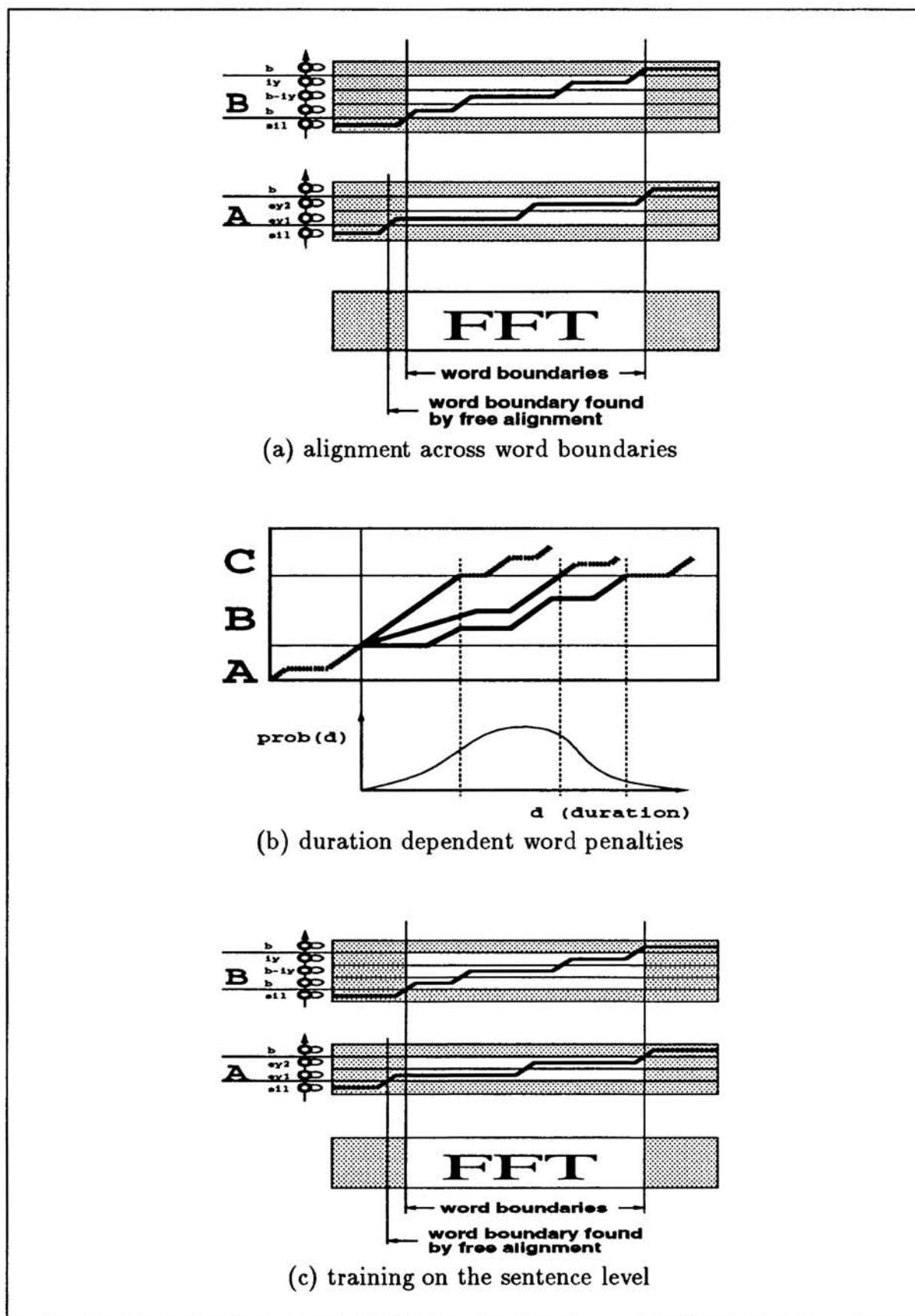

(a) alignment across word boundaries

(b) duration dependent word penalties

(c) training on the sentence level

Figure 2: Various techniques to improve sentence level recognition performance

in the same way as the phoneme boundaries within a word. Figure 2(a) shows an example in which the word to recognize is surrounded by a silence and a 'B', thus the left and right context (for all words to be recognized) is the phoneme 'sil' and 'b', respectively. The gray shaded area indicates the extension necessary to the DTW alignment. The diagram shows how a new boundary for the beginning of the word 'A' is found. As indicated in figure 3, this techniques improves continuous recognition significantly, but it doesn't help for excerpted words.

## 2.2   WORD DURATION DEPENDENT PENALIZING OF INSERTION AND DELETION ERRORS

In "continuous testing mode", instead of looking at word units the well-known "One Stage DTW" algorithm [Ney84] is used to find an optimal path through an unspecified sequence of words. The short and confusable English letters cause many word insertion and deletion errors, such as "T E" vs. "T" or "O" vs. "O O", therefore proper duration modeling is essential.

As suggested in [HW92], minimum phoneme duration can be enforced by "state duplication". In addition, we are modeling a duration and word dependent penalty $Pen_w(d) = log(k + prob_w(d))$, where the pdf $prob_w(d)$ is approximated from the training data and $k$ is a small constant to avoid zero probabilities. $Pen_w(d)$ is added to the accumulated score $AS$ of the search path, $AS = AS + \lambda_w * Pen_w(d)$, whenever it crosses the boundary of a word $w$ in Ney's "One Stage DTW" algorithm, as indicated in figure 2(b). The ratio $\lambda_w$, which determines the degree of influence of the duration penalty, is another important degree of freedom. There is no straightforward mathematically exact way to compute the effect of a change of the "weight" $\lambda_w$ to the insertion and deletion rate. Our approach is a (pseudo) gradient descent, which changes $\lambda_w$ proportional to $E(w) = (\#ins_w - \#del_w)/\#w$, i.e. we are trying to maximize the relative balance of insertion and deletion errors.

## 2.3   ERROR BACKPOPAGATION AT THE SENTENCE LEVEL

Usually the MS-TDNN is trained to classify excerpted words, but evaluated on continuously spoken sentences. We propose a simple but effective method to extend training on the sentence level. Figure 2(c) shows the alignment path of the sentence "C A B", in which a typical error, the insertion of an 'A', occurred. In a forced alignment mode (i.e. the correct sequence of words is enforced), positive training is applied along the correct path, while the units along the incorrect path receive negative training. Note that the effect of positive and negative training is neutralized if the paths are the same, only differing parts receive non-zero error backpropagation.

## 2.4   LEARNING CURVES

Figure 3 demonstrates the effect of the various training phases. The system is bootstrapped (a) during iteration 1 to 130. Word level training starts (b) at iteration 110. Word level training with additional "training across word boundaries" (c) is started at iteration 260. Excerpted word performance is not improved after (c), but continuous recognition becomes significantly better, compare (d) and (e). In (d), sentence level training is started directly after iteration 260, while in (e) sentence level training is started after additional "across boundaries (word level) training".

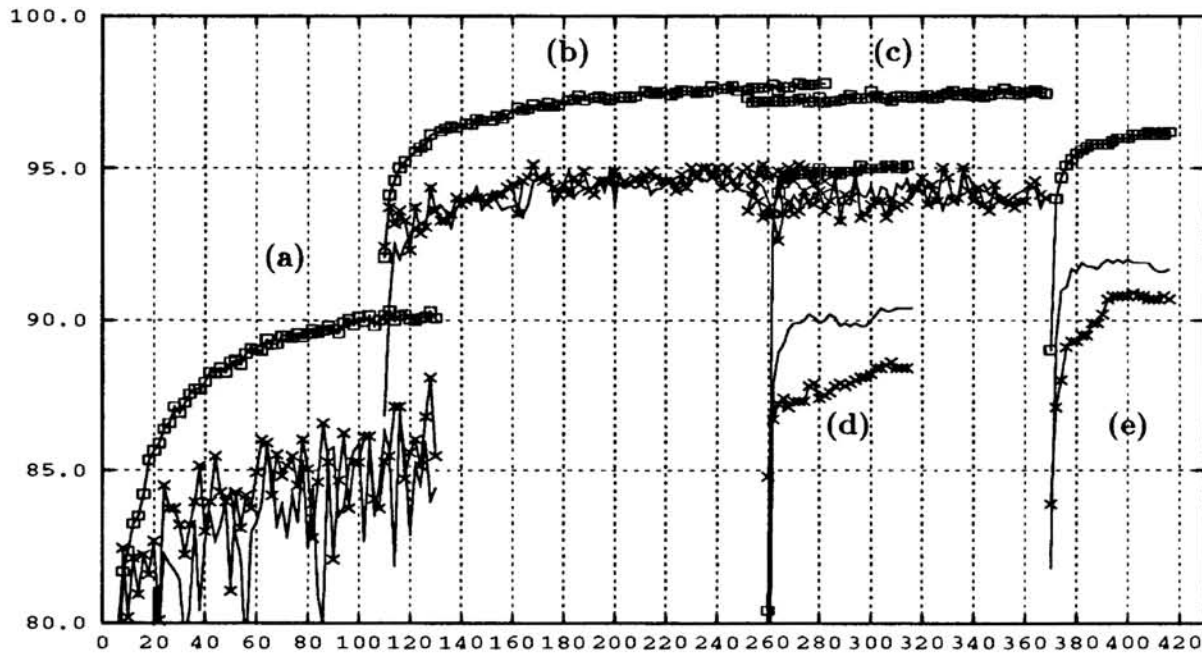

Figure 3: Learning curves (a = bootstrapping, b,c = word level (excerpted words), d,e = sentence level training (continuous speech)) on the training (□), crossvalidation (-) and test set (x) for the speaker-independent RM Spell-Mode data.

## 3   GENDER SPECIFIC SUBNETS

A straightforward approach to building a more specialized system is simply to train two entirely individual networks for male and female speakers only. During training, the gender of a speaker is known, during testing it is determined by an additional "gender identification network", which is simply another MS-TDNN with two output units representing male and female speakers. Given a sentence as input, this network classifies the speaker's gender with approx. 99% correct. The overall modularized network improved the word accuracy from 90.8% (for the "pooled" net, see table 1) to 91.3%. However, a hybrid approach with specialized gender-specific connections at the lower, input level and shared connections for the remaining net worked even better. As depicted in figure 4, in this architecture the gender identification network selects one of the two gender-specific bundles of connections between the input and hidden layer. This technique improved the word accuracy to 92.0%. More experiments with speaker-specific subnetworks are reported in [HW93].

## 4   EXPERIMENTAL RESULTS

Our MS-TDNN achieved excellent performance on both speaker dependent and independent tasks. For **speaker dependent** testing, we used the "CMU Alph-Data", with 1000 sentences (i.e. a continuously spelled string of letters) from each of 3 male and 3 female speakers. 500, 100, and 400 sentences were used as train-

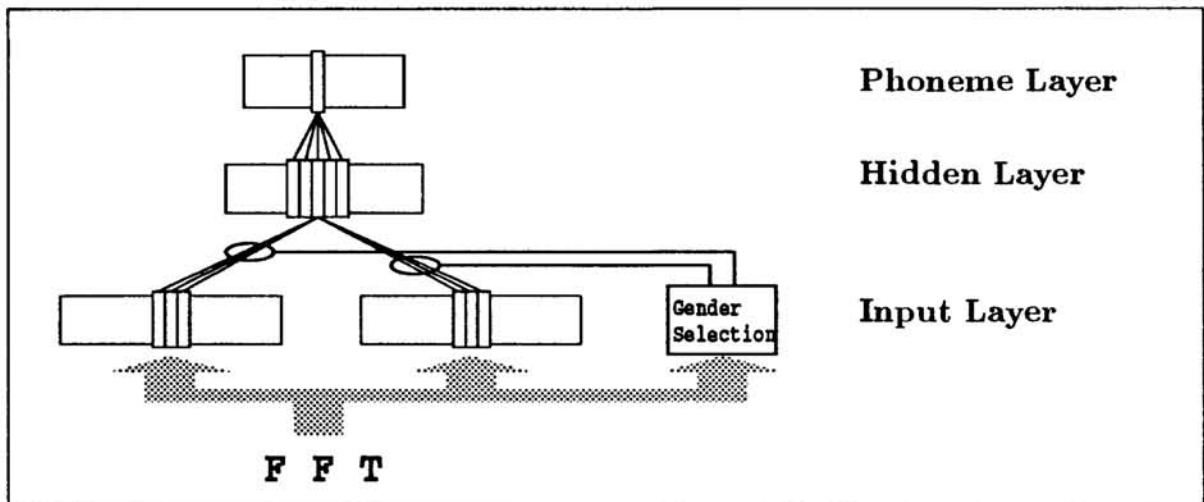

Figure 4: A network architecture with gender-specific and shared connections. Only the front-end TDNN is shown.

ing, cross-validation and test set, respectively. The DARPA Resource Management Spell-Mode Data were used for **speaker independent** testing. This data base contains about 1700 sentences, spelled by 85 male and 35 female speakers. The speech of 7 male and 4 female speakers was set aside for the test set, one sentence from all 109 and all sentences from 6 training speakers were used for crossvalidation. Table 1 summarizes our results. With the help of the training techniques described above we were able to outperform previously reported [HFW91] speaker dependent results as well as the HMM-based SPHINX System.

## 5  SUMMARY AND FUTURE WORK

We have presented a connectionist speech recognition system for high accuracy connected letter recognition. New training techniques aimed at improving sentence level recognition enabled our MS-TDNN to outperform previous systems of its own kind as well as a state-of-the art HMM-based system (SPHINX). Beyond the gender specific subnets, we are experimenting with an MS-TDNN which maintains several "internal speaker models" for a more sophisticated speaker-independent system. In the future we will also experiment with context dependent phoneme models.

**Acknowledgements**

The authors gratefully acknowledge support by the National Science Foundation and DARPA. We wish to thank Joe Tebelskis for insightful discussions, Arthur McNair for keeping our machines running, and especially Patrick Haffner. Many of the ideas presented have been developed in collaboration with him.

## Footnotes

[1]In our context, a "word" consists of one spelled letter, and a "sentence" is a continuously spelled string of letters.

## References

[CFGJ91]    R. A. Cole, M. Fanty, Gopalakrishnan, and R. D.T. Janssen. Speaker-Independent Name Retrival from Spellings Using a Database of 50,000 Names.


| Speaker Dependent (CMU Alph Data) | | | |
| 500/2500 train, 100/500 crossvalidation, 400/2000 test sentences/words | | | |
| speaker | SPHINX[HFW91] | MS-TDNN[HFW91] | our MS-TDNN |
|---|---|---|---|
| mjmt | 96.0 | 97.5 | 98.5 |
| mdbs | 83.9 | 89.7 | 91.1 |
| maem | – | – | 94.6 |
| fcaw | – | – | 98.8 |
| flgt | – | – | 86.9 |
| fee | – | – | 91.0 |
| Speaker Independent (Resource Management Spell-Mode) | | | |
| 109 (ca. 11000) train, 11 (ca. 900) test speaker (words). | | | |
| SPHINX[HH92] | | our MS-TDNN | |
|  | + Senone |  | gender specific |
| 88.7 | 90.4 | 90.8 | 92.0 |

Table 1: Word accuracy (in % on the test sets) on speaker dependent and speaker independent connected letter tasks.

In *Proceedings of the International Conference on Acoustics, Speech and Signal Processing*, Toronto, Ontario, Canada, May 1991. IEEE.

[FC90]     M. Fanty and R. Cole. Spoken letter recognition. In *Proceedings of the Neural Information Processing Systems Conference NIPS*, Denver, November 1990.

[Haf92]    P. Haffner. Connectionist Word-Level Classification in Speech Recognition. In *Proc. IEEE International Conference on Acoustics, Speech, and Signal Processing*. IEEE, 1992.

[HFW91]    P. Haffner, M. Franzini, and A. Waibel. Integrating Time Alignment and Neural Networks for High Performance Continuous Speech Recognition. In *Proc. Int. Conf. on Acoustics, Speech, and Signal Processing*. IEEE, 1991.

[HH92]     M.-Y. Hwang and X. Huang. Subphonetic Modeling with Markov States - Senone. In *Proc. IEEE International Conference on Acoustics, Speech, and Signal Processing*, pages I33 – I37. IEEE, 1992.

[HW90]     J. Hampshire and A. Waibel. A Novel Objective Function for Improved Phoneme Recognition Using Time Delay Neural Networks. *IEEE Transactions on Neural Networks*, June 1990.

[HW92]     P. Haffner and A. Waibel. Multi-state Time Delay Neural Networks for Continuous Speech Recognition. In *NIPS(4)*. Morgan Kaufman, 1992.

[HW93]     H. Hild and A. Waibel. Multi-Speaker/Speaker-Independent Architectures for the Multi-State Time Delay Neural Network. In *Proc. IEEE International Conference on Acoustics, Speech, and Signal Processing*. IEEE, 1993.

[JFC90]    R.D.T. Jansen, M. Fanty, and R. A. Cole. Speaker-independent Phonetic Classification in Continuous English Letters. In *Proceedings of the IJCNN 90, Washington D.C.*, July 1990.

[Ney84]    H. Ney. The Use of a One-Stage Dynamic Programming Algorithm for Connected Word Recognition. In *IEEE Transactions on Acoustics, Speech, and Signal Processing*, pages 263–271. IEEE, April 1984.

[WHH+89]   A. Waibel, T. Hanazawa, G. Hinton, K. Shikano, and K. Lang. Phoneme Recognition Using Time-Delay Neural Networks. *IEEE, Transactions on Acoustics, Speech and Signal Processing*, March 1989.
